# Using the Forest to See the Trees: A Graphical Model Relating Features, Objects, and Scenes

**Kevin Murphy**
MIT AI lab
Cambridge, MA 02139
murphyk@ai.mit.edu

**Antonio Torralba**
MIT AI lab
Cambridge, MA 02139
torralba@ai.mit.edu

**William T. Freeman**
MIT AI lab
Cambridge, MA 02139
wtf@ai.mit.edu

## Abstract

Standard approaches to object detection focus on local patches of the image, and try to classify them as background or not. We propose to use the *scene context* (image as a whole) as an extra source of (global) information, to help resolve local ambiguities. We present a conditional random field for jointly solving the tasks of object detection and scene classification.

## 1 Introduction

Standard approaches to object detection (e.g., [24, 15]) usually look at local pieces of the image in isolation when deciding if the object is present or not at a particular location/ scale. However, this approach may fail if the image is of low quality (e.g., [23]), or the object is too small, or the object is partly occluded, etc. In this paper we propose to use the image as a whole as an extra global feature, to help overcome local ambiguities.

There is some psychological evidence that people perform rapid global scene analysis before conducting more detailed local object analysis [4, 2]. The key computational question is how to represent the whole image in a compact, yet informative, form. [21] suggests a representation, called the "gist" of the image, based on PCA of a set of spatially averaged filter-bank outputs. The gist acts as an holistic, low-dimensional representation of the whole image. They show that this is sufficient to provide a useful prior for what types of objects may appear in the image, and at which locations/scale.

We extend [21] by combining the prior suggested by the gist with the outputs of bottom-up, local object detectors, which are trained using boosting (see Section 2). Note that this is quite different from approaches that use joint spatial constraints between the locations of objects, such as [11, 20, 19, 8]. In our case, the spatial constraints come from the image as a whole, not from other objects. This is computationally much simpler.

Another task of interest is detecting if the object is present anywhere in the image, regardless of location. (This can be useful for object-based image retrieval.) In principle, this is straightforward: we declare the object is present iff the detector fires (at least once) at any location/scale. However, this means that a single false positive at the patch level can cause a 100% error rate at the image level. As we will see in Section 4, even very good detectors can perform poorly at this task. The gist, however, is able to perform quite well at suggesting the presence of types of objects, without using a detector at all. In fact, we can

use the gist to decide if it is even "worth" running a detector, although we do not explore this here.

Often, the presence of certains types of objects is correlated, e.g., if you see a keyboard, you expect to see a screen. Rather than model this correlation directly, we introduce a hidden common cause/ factor, which we call the "scene". In Section 5, we show how we can reliably determine the type of scene (e.g., office, corridor or street) using the gist. Scenes can also be defined in terms of the objects which are present in the image. Hence we combine the tasks of scene classification and object-presence detection using a tree-structured graphical model: see Section 6. We perform top-down inference (scenes to objects) and bottom-up inference (objects to scenes) in this model. Finally, we conclude in Section 7. (Note: there is a longer, online version of this paper available at www.ai.mit.edu/~murphyk/Papers/nips2003_long.pdf, which has more details and experimental results than could fit into 8 pages.)

## 2 Object detection and localization

For object detection there are at least three families of approaches: parts-based (an object is defined as a specific spatial arrangement of small parts e.g., [6]), patch-based (we classify each rectangular image region as object or background), and region-based (a region of the image is segmented from the background and is described by a set of features that provide texture and shape information e.g., [5]).

Here we use a patch-based approach. For objects with rigid, well-defined shapes (screens, keyboards, people, cars), a patch usually contains the full object and a small portion of the background. For the rest of the objects (desks, bookshelves, buildings), rectangular patches may contain only a piece of the object. In that case, the region covered by a number of patches defines the object. In such a case, the object detector will rely mostly on the textural properties of the patch.

The main advantage of the patch-based approach is that object-detection can be reduced to a binary classification problem. Specifically, we compute $P(O_i^c = 1|v_i^c)$ for each class $c$ and patch $i$ (ranging over location and scale), where $O_i^c = 1$ if patch $i$ contains (part of) an instance of class $c$, and $O_i^c = 0$ otherwise; $v_i^c$ is the feature vector (to be described below) for patch $i$ computed for class $c$.

To detect an object, we slide our detector across the image pyramid and classify all the patches at each location and scale (20% increments of size and every other pixel in location). After performing non-maximal suppression [1], we report as detections all locations for which $P(O_i^c|v_i^c)$ is above a threshold, chosen to given a desired trade-off between false positives and missed detections.

### 2.1 Features for objects and scenes

We would like to use the same set of features for detecting a variety of object types, as well as for classifying scenes. Hence we will create a large set of features and use a feature selection algorithm (Section 2.2) to select the most discriminative subset.

We compute a single feature $k$ for image patch $i$ in three steps, as follows. First we convolve the (monochrome) patch $I_i(x)$ with a filter $g_k(x)$, chosen from the set of 13 (zero-mean) filters shown in Figure 1(a). This set includes oriented edges, a Laplacian filter, corner detectors and long edge detectors. These features can be computed efficiently: The filters used can be obtained by convolution of 1D filters (for instance, the long edge filters are obtained by the convolution of the two filters $[-1\ 0\ 1]^T$ and $[1\ 1\ 1\ 1\ 1]$) or as linear combinations of the other filter outputs (e.g., the first six filters are steerable).

We can summarize the response of the patch convolved with the filter, $|I_i(x)*g_k(x)|$, using a histogram. For natural images, we can further summarize this histogram using just two statistics, the variance and the kurtosis [7]. Hence in step two, we compute $|I_i(x)*g_k(x)|^{\gamma_k}$, for $\gamma_k \in \{2,4\}$. (The kurtosis is useful for characterizing texture-like regions.)

Often we are only interested in the response of the filter within a certain region of the patch. Hence we can apply one of 30 different spatial templates, which are shown in Figure 1(b). The use of a spatial template provides a crude encoding of "shape" inside the rectangular patch. We use rectangular masks because we can efficiently compute the average response of a filter within each region using the integral image [24].[1]

Summarizing, we can compute feature $k$ for patch $i$ as follows: $f_i(k) = \sum_x w_k(x) \left(|I(x)*g_k(x)|^{\gamma_k}\right)_i$. (To achieve some illumination invariance, we also standardize each feature vector on a per-patch basis.) The feature vector has size $13 \times 30 \times 2 = 780$ (the factor of 2 arises because we consider $\gamma_k = 2$ or 4).

Figure 2 shows some of the features selected by the learning algorithm for different kinds of objects. For example, we see that computer monitor screens are characterized by long horizontal or vertical lines on the edges of the patch, whereas buildings, seen from the outside, are characterized by cross-like texture, due to the repetitive pattern of windows.

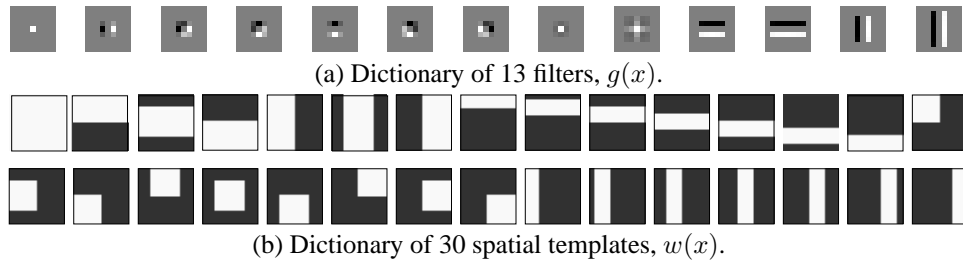

(a) Dictionary of 13 filters, $g(x)$.

(b) Dictionary of 30 spatial templates, $w(x)$.

Figure 1: **(a) Dictionary of filters**. Filter 1 is a delta function, 2–7 are 3x3 Gaussian derivatives, 8 is a 3x3 Laplacian, 9 is a 5x5 corner detector, 10–13 are long edge detectors (of size 3x5, 3x7, 5x3 and 7x3). **(b) Dictionary of 30 spatial templates**. Template 1 is the whole patch, 2–7 are all sub-patches of size 1/2, 8–30 are all sub-patches of size 1/3.

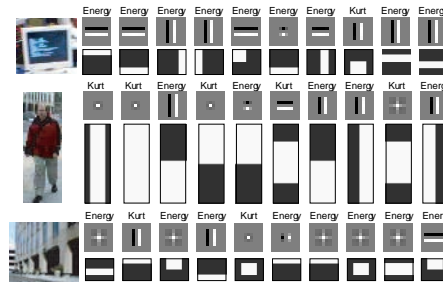

Figure 2: Some of the features chosen after 100 rounds of boosting for recognizing screens, pedestrians and buildings. Features are sorted in order of decreasing weight, which is a rough indication of importance. "Energy" means $\gamma_k = 2$ and "Kurt" (kurtosis) means $\gamma_k = 4$.

## 2.2 Classifier

Following [24], our detectors are based on a classifier trained using boosting. There are many variants of boosting [10, 9, 17], which differ in the loss function they are trying to optimize, and in the gradient directions which they follow. We, and others [14], have found that GentleBoost [10] gives higher performance than AdaBoost [17], and requires fewer iterations to train, so this is the version we shall (briefly) present below.

The boosting procedure learns a (possibly weighted) combination of base classifiers, or "weak learners": $\alpha(v) = \sum_t \alpha_t h_t(v)$, where $v$ is the feature vector of the patch, $h_t$ is the base classifier used at round $t$, and $\alpha_t$ is its corresponding weight. (GentleBoost, unlike AdaBoost, does not weight the outputs of the weak learners, so $\alpha_t = 1$.) For the weak classifiers we use regression stumps of the form $h(v) = a[v_f > \theta] + b$, where $[v_f > \theta] = 1$ iff component $f$ of the feature vector $v$ is above threshold $\theta$. For most of the objects we used about 100 rounds of boosting. (We use a hold-out set to monitor overfitting.) See Figure 2 for some examples of the selected features.

The output of a boosted classifier is a "confidence-rated prediction", $\alpha$. We convert this to a probability using logistic regression: $P(O_i^c = 1 | \alpha(v_i^c)) = \sigma(w^T[1 \ \alpha])$, where $\sigma(x) = 1/(1 + \exp(-x))$ is the sigmoid function [16]. We can then change the hit rate/false alarm rate of the detector by varying the threshold on $P(O = 1 | \alpha)$.

Figure 3 summarizes the performances of the detectors for a set of objects on isolated patches (not whole images) taken from the test set. The results vary in quality since some objects are harder to recognize than others, and because some objects have less training data. When we trained and tested our detector on the training/testing sets of side-views of cars from UIUC[2], we outperformed the detector of [1] at every point on the precision-recall curve (results not shown), suggesting that our base-line detectors can match state-of-the-art detectors when given enough training data.

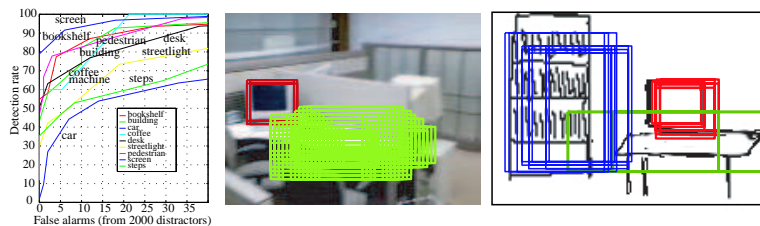

Figure 3: a) ROC curves for 9 objects; we plot hit rate vs number of false alarms, when the detectors are run on isolated test patches. b) Example of the detector output on one of the test set images, before non-maximal suppression. c) Example of the detector output on a line drawing of a typical office scene. The system correctly detects the screen, the desk and the bookshelf.

## 3 Improving object localization by using the gist

One way to improve the speed and accuracy of a detector is to reduce the search space, by only running the detector in locations/ scales that we expect to find the object. The expected location/scale can be computed on a per image basis using the gist, as we explain below. (Thus our approach is more sophisticated than having a fixed prior, such as "keyboards always occur in the bottom half of an image".)

If we only run our detectors in a predicted region, we risk missing objects. Instead, we run our detectors everywhere, but we penalize detections that are far from the predicted

location/scale. Thus objects in unusual locations have to be particularly salient (strong local detection score) in order to be detected, which accords with psychophysical results of human observers.

We define the gist as a feature vector summarizing the whole image, and denote it by $v_G$. One way to compute this is to treat the whole image as a single patch, and to compute a feature vector for it as described in Section 2.1. If we use 4 image scales and 7 spatial masks, the gist will have size $13 \times 7 \times 2 \times 4 = 728$. Even this is too large for some methods, so we consider another variant that reduces dimensionality further by using PCA on the gist-minus-kurtosis vectors. Following [22, 21], we take the first 80 principal components; we call this the PCA-gist.

We can predict the expected location/scale of objects of class $c$ given the gist, $E[X^c|v^G]$, by using a regression procedure. We have tried linear regression, boosted regression [9], and cluster-weighted regression [21]; all approaches work about equally well.

Using the gist it is easy to distinguish long-distance from close-up shots (since the overall structure of the image looks quite different), and hence we might predict that the object is small or large respectively. We can also predict the expected height. However, we cannot predict the expected horizontal location, since this is typically unconstrained by the scene.

To combine the local and global sources of information, we construct a feature vector $f$ which combines the output of the boosted detector, $\alpha(v_i^c)$, and the vector between the location of the patch and the predicted location for objects of this class, $x_i^c - \hat{x}^c$. We then train another classifier to compute $P(O_i^c = 1|f(\alpha(v_i^c), x_i^c, \hat{x}^c))$ using either boosting or logistic regression. In Figure 4, we compare localization performance using just the detectors, $P(O_i^c = 1|\alpha(v_i^c))$, and using the detectors and the predicted location, $P(O_i^c = 1|f(\alpha(v_i^c), x_i^c, \hat{x}^c))$. For keyboards (which are hard to detect) we see that using the predicted location helps a lot, whereas for screens (which are easy to detect), the location information does not help.

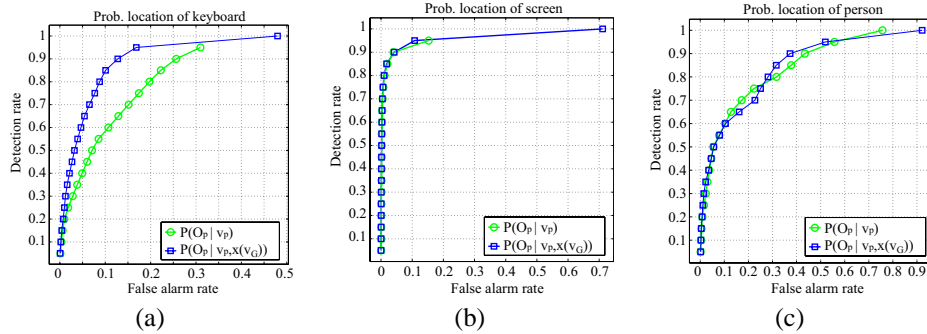

Figure 4: ROC curves for detecting the location of objects in the image: (a) keyboard, (b) screen, (c) person. The green circles are the local detectors alone, and the blue squares are the detectors and predicted location.

## 4  Object presence detection

We can compute the probability that the object exists anywhere in the image (which can be used for e.g., object-based image retrieval) by taking the OR of all the detectors:

$$P(E^c = 1|v_{1:N}^c) = \vee_i P(O^c = 1|v_{1:N}^c).$$

Unfortunately, this leads to massive overconfidence, since the patches are not independent. As a simple approximation, we can use

$$P(E^c = 1|v_{1:N}^c) \approx \max_i P(E^c = 1|v_{1:N}^c) = P(E^c = 1| \max_i \alpha_i(v_i^c)) = P(E^c = 1|\alpha_{max}^c).$$

Unfortunately, even for good detectors, this can give poor results: the probability of error at the image level is $1 - \prod_i(1 - q_i) = 1 - (1 - q)^N$, where $q$ is the probability of error at the patch level and $N$ is the number of patches. For a detector with a reasonably low false alarm rate, say $q = 10^{-4}$, and $N = 5000$ patches, this gives a 40% false detection rate at the image level! For example, see the reduced performance at the image level of the screen detector (Figure 5(a)), which performs very well at the patch level (Figure 4(a)).

An alternative approach is to use the gist to predict the presence of the object, without using a detector at all. This is possible because the overall structure of the image can suggest what kind of scene this is (see Section 5), and this in turn suggests what kinds of objects are present (see Section 6). We trained another boosted classifier to predict $P(E^c = 1|v^G)$; results are shown Figure 5. For poor detectors, such as keyboards, the gist does a much better job than the detectors, whereas for good detectors, such as screens, the results are comparable. Finally, we can combine both approaches by constructing a feature vector from the output of the global and local boosted classifiers and using logistic regression:

$$P(E^c = 1|v^G, v_{1:N}^c) = \sigma(w^T[1 \;\; \alpha(v^G)\alpha_{max}^c]).$$

However, this seems to offer little improvement over the gist alone (see Figure 5), presumably because our detectors are not very good.

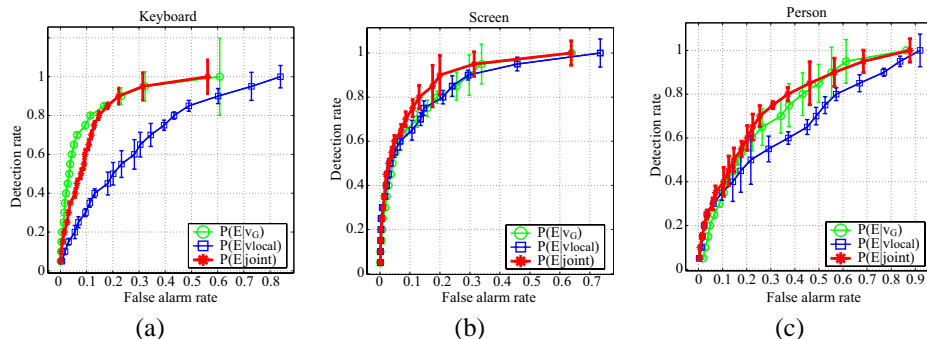

Figure 5: ROC curves for detecting the presence of object classes in the image: (a) keyboard (b) screen, (c) person. The green circles use the gist alone, the blue squares use the detectors alone, and the red stars use the joint model, which uses the gist and all the detectors from all the object classes.

# 5    Scene classification

As mentioned in the introduction, the presence of many types of objects is correlated. Rather than model this correlation directly, we introduce a latent common "cause", which we call the "scene". We assume that object presence is conditionally independent given the scene, as explained in Section 6. But first we explain how we recognize the scene type, which in this paper can be office, corridor or street.

The approach we take to scene classification is simple. We train a one-vs-all binary classifier for recognizing each type of scene using boosting applied to the gist.[3]   Then we

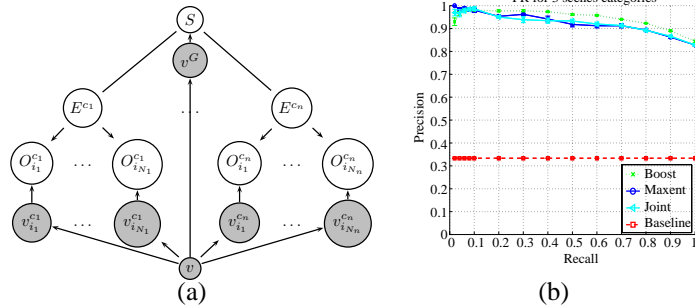

(a)                                      (b)

Figure 6: **(a)** Graphical model for scene and object recognition. $n = 6$ is the number of object classes, $N_c \sim 5000$ is the number of patches for class $c$. Other terms are defined in the text. **(b)** Precision-recall curve for scene classification.

normalize the results: $P(S = s|v^G) = \frac{P(S^s=1|v^G)}{\sum_{s'} P(S^{s'}=1|v^G)}$ where $P(S^s = 1|v^G)$ is the output of the s-vs-other classifier.[4]

## 6  Joint scene classification and object-presence detection

We now discuss how we can use scene classification to facilitate object-presence detection, and vice versa. The approach is based on the tree-structured graphical model[5] in Figure 6(a), which encodes our assumption that the objects are conditionally independent given the scene.

This graphical model encodes the following conditional joint density:

$$P(S, E^{1:n}, O_{1:N}^c, \ldots, O_{1:N}^{c_n}|v) = \frac{1}{Z} P(S|v^G) \prod_c \phi(E^c, S) \prod_i P(O_i^c|E^c, v_i^c)$$

where $v^G$ and $v_i^c$ are deterministic functions of the image $v$ and $Z$ is a normalizing constant. called the partition function (which is tracatable to compute, since the graph is a tree). By conditioning on the observations as opposed to generating them, we are free to incorporate arbitrary, possibly overlapping features (local and global), without having to make strong independence assumptions c.f., [13, 12].

We now define the individual terms in this expression. $P(S|v^G)$ is the output of boosting as described in Section 5. $\phi(E^c, S)$ is essentially a table which counts the number of times object type $c$ occurs in scene type $S$. Finally, we define

$$P(O_i^c = 1|E^c = e, v_i^c) = \begin{cases} \sigma(w^T[1 \ \alpha(v_i^c)]) & \text{if } e = 1 \\ 0 & \text{if } e = 0 \end{cases}$$

This means that if we know the object is absent in the image ($E^c = 0$), then all the local detectors should be turned off ($O_i^c = 0$); but if the object is present ($E^c = 1$), we do not know where, so we allow the local evidence, $v_i^c$, to decide which detectors should turn on. We can find the maximum likelihood estimates of the parameters of this model by training it jointly using a gradient procedure; see the long version of this paper for details.

In Figure 5, we see that we can reliably detect the presence of the object in an image without using the gist directly, providing we

know what the scene type is (the red curve, derived from the joint model in this section, is basically the same as the green curve, derived from the gist model in Section 4). The importance of this is that it is easy to label images with their scene type, and hence to train $P(S|v^G)$, but it is much more time consuming to annotate objects, which is required to train $P(E^c|v^G)$.[6]

## 7 Conclusions and future work

We have shown how to combine global and local image features to solve the tasks of object detection and scene recognition. In the future, we plan to try a larger number of object classes. Also, we would like to investigate methods for choosing which order to run the detectors. For example, one can imagine a scenario in which we run the screen detector first (since it is very reliable); if we discover a screen, we conclude we are in an office, and then decide to look for keyboards and chairs; but if we don't discover a screen, we might be in a corridor or a street, so we choose to run another detector to disambiguate our belief state. This corresponds to a dynamic message passing protocol on the graphical model.

## Footnotes

[1] The Viola and Jones [24] feature set is equivalent to using these masks plus a delta function filter; the result is like a Haar wavelet basis. This has the advantage that objects of any size can be detected without needing an image pyramid, making the system very fast. By contrast, since our filters have fixed spatial support, we need to down-sample the image to detect large objects.

[2]http://l2r.cs.uiuc.edu/~cogcomp/Data/Car/

[3]An alternative would be to use the multi-class LogitBoost algorithm [10]. However, training separate one-vs-all classifiers allows them to have different internal structure (e.g., number of rounds).

[4]For scenes, it is arguably more natural to allow multiple labels, as in [3], rather than forcing each scene into a single category; this can be handled with a simple modification of boosting [18].

[5]The graph is a tree once we remove the observed nodes.

[6]Since we do not need to know the location of the object in the image in order to train $P(E^c|v^G)$, we can use partially annotated data such as image captions, as used in [5].

## References

[1] S. Agarwal and D. Roth. Learning a sparse representation for object detection. In *Proc. European Conf. on Computer Vision*, 2002.

[2] I. Biederman. On the semantics of a glance at a scene. In M. Kubovy and J. Pomerantz, editors, *Perceptual organization*, pages 213–253. Erlbaum, 1981.

[3] M. Boutell, X. Shen, J. Luo, and C. Brown. Multi-label semantic scene classification. Technical report, Dept. Comp. Sci. U. Rochester, 2003.

[4] D. Davon. Forest before the trees: the precedence of global features in visual perception. *Cognitive Psychology*, 9:353–383, 1977.

[5] Pinary Duygulu, Kobus Barnard, Nando de Freitas, David Forsyth, and Michael I. Jordan. Object recognition as machine translation: Learning a lexicon for a fixed image vocabulary. In *Proc. European Conf. on Computer Vision*, 2002.

[6] R. Fergus, P. Perona, and A. Zisserman. Object class recognition by unsupervised scale-invariant learning. In *Proc. IEEE Conf. Computer Vision and Pattern Recognition*, 2003.

[7] D. Field. Relations between the statistics of natural images and the response properties of cortical cells. *J. Opt. Soc. Am.*, A4:2379–2394, 1987.

[8] M. Fink and P. Perona. Mutual boosting for contextual influence. In *Advances in Neural Info. Proc. Systems*, 2003.

[9] J. Friedman. Greedy function approximation: a gradient boosting machine. *Annals of Statistics*, 29:1189–1232, 2001.

[10] J. Friedman, T. Hastie, and R. Tibshirani. Additive logistic regression: a statistical view of boosting. *Annals of statistics*, 28(2):337–374, 2000.

[11] R. Haralick. Decision making in context. *IEEE Trans. on Pattern Analysis and Machine Intelligence*, 5:417–428, 1983.

[12] Sanjiv Kumar and Martial Hebert. Discriminative random fields: A discriminative framework for contextual interaction in classification. In *IEEE Conf. on Computer Vision and Pattern Recognition*, 2003.

[13] J. Lafferty, A. McCallum, and F. Pereira. Conditional random fields: Probabilistic models for segmenting and labeling sequence data. In *Intl. Conf. on Machine Learning*, 2001.

[14] R. Lienhart, A. Kuranov, and V. Pisarevsky. Empirical analysis of detection cascades of boosted classifiers for rapid object detection. In *DAGM 25th Pattern Recognition Symposium*, 2003.

[15] C. Papageorgiou and T. Poggio. A trainable system for object detection. *Intl. J. Computer Vision*, 38(1):15–33, 2000.

[16] J. Platt. Probabilistic outputs for support vector machines and comparisons to regularized likelihood methods. In A. Smola, P. Bartlett, B. Schoelkopf, and D. Schuurmans, editors, *Advances in Large Margin Classifiers*. MIT Press, 1999.

[17] R. Schapire. The boosting approach to machine learning: An overview. In *MSRI Workshop on Nonlinear Estimation and Classification*, 2001.

[18] Robert E. Schapire and Yoram Singer. BoosTexter: A boosting-based system for text categorization. *Machine Learning*, 39(2/3):135–168, 2000.

[19] A. Singhal, Jiebo Luo, and Weiyu Zhu. Probabilistic spatial context models for scene content understanding. In *Proc. IEEE Conf. Computer Vision and Pattern Recognition*, 2003.

[20] T. M. Strat and M. A. Fischler. Context-based vision: recognizing objects using information from both 2-D and 3-D imagery. *IEEE Trans. on Pattern Analysis and Machine Intelligence*, 13(10):1050–1065, 1991.

[21] A. Torralba. Contextual priming for object detection. *Intl. J. Computer Vision*, 53(2):153–167, 2003.

[22] A. Torralba, K. Murphy, W. Freeman, and M. Rubin. Context-based vision system for place and object recognition. In *Intl. Conf. Computer Vision*, 2003.

[23] A. Torralba and P. Sinha. Detecting faces in impoverished images. Technical Report 028, MIT AI Lab, 2001.

[24] Paul Viola and Michael Jones. Robust real-time object detection. *International Journal of Computer Vision - to appear*, 2002.

